# Select and Sample — A Model of Efficient Neural Inference and Learning

**Jacquelyn A. Shelton,    Jörg Bornschein,    Abdul-Saboor Sheikh**
Frankfurt Institute for Advanced Studies
Goethe-University Frankfurt, Germany
`{shelton,bornschein,sheikh}@fias.uni-frankfurt.de`

| | |
|---|---|
| **Pietro Berkes** | **Jörg Lücke** |
| Volen Center for Complex Systems | Frankfurt Institute for Advanced Studies |
| Brandeis University, Boston, USA | Goethe-University Frankfurt, Germany |
| `berkes@brandeis.edu` | `luecke@fias.uni-frankfurt.de` |

## Abstract

An increasing number of experimental studies indicate that perception encodes a posterior probability distribution over possible causes of sensory stimuli, which is used to act close to optimally in the environment. One outstanding difficulty with this hypothesis is that the exact posterior will in general be too complex to be represented directly, and thus neurons will have to represent an approximation of this distribution. Two influential proposals of efficient posterior representation by neural populations are: 1) neural activity represents samples of the underlying distribution, or 2) they represent a parametric representation of a variational approximation of the posterior. We show that these approaches can be combined for an inference scheme that retains the advantages of both: it is able to represent multiple modes and arbitrary correlations, a feature of sampling methods, and it reduces the represented space to regions of high probability mass, a strength of variational approximations. Neurally, the combined method can be interpreted as a feed-forward preselection of the relevant state space, followed by a neural dynamics implementation of Markov Chain Monte Carlo (MCMC) to approximate the posterior over the relevant states. We demonstrate the effectiveness and efficiency of this approach on a sparse coding model. In numerical experiments on artificial data and image patches, we compare the performance of the algorithms to that of exact EM, variational state space selection alone, MCMC alone, and the combined select and sample approach. The select and sample approach integrates the advantages of the sampling and variational approximations, and forms a robust, neurally plausible, and very efficient model of processing and learning in cortical networks. For sparse coding we show applications easily exceeding a thousand observed and a thousand hidden dimensions.

## 1   Introduction

According to the recently quite influential statistical approach to perception, our brain represents not only the most likely interpretation of a stimulus, but also its corresponding uncertainty. In other words, ideally the brain would represent the full posterior distribution over all possible interpretations of the stimulus, which is statistically optimal for inference and learning [1, 2, 3] – a hypothesis supported by an increasing number of psychophysical and electrophysiological results [4, 5, 6, 7, 8, 9].

Although it is generally accepted that humans indeed maintain a complex posterior representation, one outstanding difficulty with this approach is that the full posterior distribution is in general very complex, as it may be highly correlated (due to explaining away effects), multimodal (multiple possible interpretations), and very high-dimensional. One approach to address this problem in neural circuits is to let neuronal activity represent the parameters of a variational approximation of the real posterior [10, 11]. Although this approach can approximate the full posterior, the number of neurons explodes with the number of variables – for example, approximation via a Gaussian distribution requires $N^2$ parameters to represent the covariance matrix over $N$ variables. Another approach is to identify neurons with variables and interpret neural activity as samples from their posterior [12, 13, 3]. This interpretation is consistent with a range of experimental observations, including neural variability (which would result from the uncertainty in the posterior) and spontaneous activity (corresponding to samples from the prior in the absence of a stimulus) [3, 9]. The advantage of using sampling is that the number of neurons scales linearly with the number of variables, and it can represent arbitrarily complex posterior distributons given enough samples. The latter part is the issue: collecting a sufficient number of samples to form such a complex, high-dimensional representation is quite time-costly. Modeling studies have shown that a small number of samples are sufficient to perform well on low-dimensional tasks (intuitively, this is because taking a low-dimensional marginal of the posterior accumulates samples over all dimensions) [14, 15]. However, most sensory data is inherently very high-dimensional. As such, in order to faithfully represent visual scenes containing potentially many objects and object parts, one requires a high-dimensional latent space to represent the high number of potential causes, which returns to the problem sampling approaches face in high dimensions.

The goal of the line of research pursued here is to address the following questions: 1) can we find a sophisticated representation of the posterior for very high-dimensional hidden spaces? 2) as this goal is believed to be shared by the brain, can we find a biologically plausible solution reaching it? In this paper we propose a novel approach to approximate inference and learning that addresses the drawbacks of sampling as a neural processing model, yet maintains its beneficial posterior representation and neural plausibility. We show that sampling can be combined with a preselection of candidate units. Such a selection connects sampling to the influential models of neural processing that emphasize feed-forward processing ([16, 17] and many more), and is consistent with the popular view of neural processing and learning as an interplay between feed-forward and recurrent stages of processing [18, 19, 20, 21, 12]. Our combined approach emerges naturally by interpreting feed-forward selection and sampling as approximations to exact inference in a probabilistic framework for perception.

## 2 A Select and Sample Approach to Approximate Inference

Inference and learning in neural circuits can be regarded as the task of inferring the true hidden causes of a stimulus. An example is inferring the objects in a visual scene based on the image projected on the retina. We will refer to the sensory stimulus (the image) as a *data point*, $\vec{y} = (y_1, \ldots, y_D)$, and we will refer to the hidden causes (the objects) as $\vec{s} = (s_1, \ldots, s_H)$ with $s_h$ denoting *hidden variable* or *hidden unit* $h$. The data distribution can then be modeled by a generative data model: $p(\vec{y} \,|\, \Theta) = \sum_{\vec{s}} p(\vec{y} \,|\, \vec{s}, \Theta) \, p(\vec{s} \,|\, \Theta)$ with $\Theta$ denoting the parameters of the model[1]. If we assume that the data distribution can be optimally modeled by the generative distribution for optimal parameters $\Theta^*$, then the posterior probability $p(\vec{s} \,|\, \vec{y}, \Theta^*)$ represents optimal inference given a data point $\vec{y}$. The parameters $\Theta^*$ given a set of $N$ data points $Y = \{\vec{y}_1, \ldots, \vec{y}_N\}$ are given by the maximum likelihood parameters $\Theta^* = \text{argmax}_\Theta \{p(Y \,|\, \Theta)\}$.

A standard procedure to find the maximum likelihood solution is expectation maximization (EM). EM iteratively optimizes a lower bound of the data likelihood by inferring the posterior distribution over hidden variables given the current parameters (the E-step), and then adjusting the parameters to maximize the likelihood of the data averaged over this posterior (the M-step). The M-step updates typically depend only on a small number of expectation values of the posterior as given by

$$\langle g(\vec{s}) \rangle_{p(\vec{s} \,|\, \vec{y}^{(n)}, \Theta)} \;=\; \sum_{\vec{s}} p(\vec{s} \,|\, \vec{y}^{(n)}, \Theta) \, g(\vec{s}) \,, \tag{1}$$

where $g(\vec{s})$ is usually an elementary function of the hidden variables (e.g., $g(\vec{s}) = \vec{s}$ or $g(\vec{s}) = \vec{s}\vec{s}^T$ in the case of standard sparse coding). For any non-trivial generative model, the computation of

expectation values (1) is the computationally demanding part of EM optimization. Their exact computation is often intractable and many well-known algorithms (e.g., [22, 23]) rely on estimations. The EM iterations can be associated with neural processing by the assumption that neural activity represents the posterior over hidden variables (E-step), and that synaptic plasticity implements changes to model parameters (M-step). Here we will consider two prominent models of neural processing on the ground of approximations to the expectation values (1) and show how they can be combined.

**Selection.** Feed-forward processing has frequently been discussed as an important component of neural processing [16, 24, 17, 25]. One perspective on this early component of neural activity is as a preselection of candidate units or hypotheses for a given sensory stimulus ([18, 21, 26, 19] and many more), with the goal of reducing the computational demand of an otherwise too complex computation. In the context of probabilistic approaches, it has recently been shown that preselection can be formulated as a variational approximation to exact inference [27]. The variational distribution in this case is given by a truncated sum over possible hidden states:

$$p(\vec{s}\,|\,\vec{y}^{(n)}, \Theta) \approx q_n(\vec{s}; \Theta) = \frac{p(\vec{s}\,|\,\vec{y}^{(n)}, \Theta)}{\sum\limits_{\vec{s}' \in \mathcal{K}_n} p(\vec{s}'\,|\,\vec{y}^{(n)}, \Theta)}\, \delta(\vec{s} \in \mathcal{K}_n) = \frac{p(\vec{s}, \vec{y}^{(n)}\,|\,\Theta)}{\sum\limits_{\vec{s}' \in \mathcal{K}_n} p(\vec{s}', \vec{y}^{(n)}\,|\,\Theta)}\, \delta(\vec{s} \in \mathcal{K}_n) \quad (2)$$

where $\delta(\vec{s} \in \mathcal{K}_n) = 1$ if $\vec{s} \in \mathcal{K}_n$ and zero otherwise. The subset $\mathcal{K}_n$ represents the preselected latent states. Given a data point $\vec{y}^{(n)}$, Eqn. 2 results in good approximations to the posterior if $\mathcal{K}_n$ contains most posterior mass. Since for many applications the posterior mass is concentrated in small volumes of the state space, the approximation quality can stay high even for relatively small sets $\mathcal{K}_n$. This approximation can be used to compute efficiently the expectation values needed in the M-step (1):

$$\langle g(\vec{s}) \rangle_{p(\vec{s}\,|\,\vec{y}^{(n)}, \Theta)} \approx \langle g(\vec{s}) \rangle_{q_n(\vec{s}; \Theta)} = \frac{\sum_{\vec{s} \in \mathcal{K}_n} p(\vec{s}, \vec{y}^{(n)}\,|\,\Theta)\, g(\vec{s})}{\sum_{\vec{s}' \in \mathcal{K}_n} p(\vec{s}', \vec{y}^{(n)}\,|\,\Theta)}\,. \quad (3)$$

Eqn. 3 represents a reduction in required computational resources as it involves only summations (or integrations) over the smaller state space $\mathcal{K}_n$. The requirement is that the set $\mathcal{K}_n$ needs to be selected prior to the computation of expectation values, and the final improvement in efficiency relies on such selections being efficiently computable. As such, a *selection function* $\mathcal{S}_h(\vec{y}, \Theta)$ needs to be carefully chosen in order to define $\mathcal{K}_n$; $\mathcal{S}_h(\vec{y}, \Theta)$ efficiently selects the candidate units $s_h$ that are most likely to have contributed to a data point $\vec{y}^{(n)}$. $\mathcal{K}_n$ can then be defined by:

$$\mathcal{K}_n = \{\vec{s}\,|\, \text{for all } h \notin \mathcal{I}:\ s_h = 0\}\,, \quad (4)$$

where $\mathcal{I}$ contains the $H'$ indices $h$ with the highest values of $\mathcal{S}_h(\vec{y}, \Theta)$ (compare Fig. 1). For sparse coding models, for instance, we can exploit that the posterior mass lies close to low dimensional subspaces to define the sets $\mathcal{K}_n$ [27, 28], and appropriate $\mathcal{S}_h(\vec{y}, \Theta)$ can be found by deriving efficiently computable upper-bounds for probabilities $p(s_h = 1\,|\,\vec{y}^{(n)}, \Theta)$ [27, 28] or by derivations based on taking limits for no data noise [27, 29]. For more complex models, see [27] (Sec. 5.3-4) for a discussion of suitable selection functions. Often the precise form of $\mathcal{S}_h(\vec{y}, \Theta)$ has limited influence on the final approximation accuracy because a) its values are not used for the approximation (3) itself and b) the size of sets $\mathcal{K}_n$ can often be chosen generously to easily contain the regions with large posterior mass. The larger $\mathcal{K}_n$ the less precise the selection has to be. For $\mathcal{K}_n$ equal to the entire state space, no selection is required and the approximations (2) and (3) fall back to the case of exact inference.

**Sampling.** An alternative way to approximate the expectation values in eq. 1 is by sampling from the posterior distribution, and using the samples to compute the average:

$$\langle g(\vec{s}) \rangle_{p(\vec{s}\,|\,\vec{y}^{(n)}, \Theta)} \approx \tfrac{1}{M} \sum_{m=1}^{M} g(\vec{s}^{(m)}) \text{ with } \vec{s}^{(m)} \sim p(\vec{s}\,|\,\vec{y}, \Theta). \quad (5)$$

The challenging aspect of this approach is to efficiently draw samples from the posterior. In a high-dimensional sample space, this is mostly done by Markov Chain Monte Carlo (MCMC). This class of methods draws samples from the posterior distribution such that each subsequent sample is drawn relative to the current state, and the resulting sequence of samples form a Markov chain. In the limit of a large number of samples, Monte Carlo methods are theoretically able to represent any probability distribution. However, the number of samples required in high-dimensional spaces can be very large (Fig. 1A, sampling).

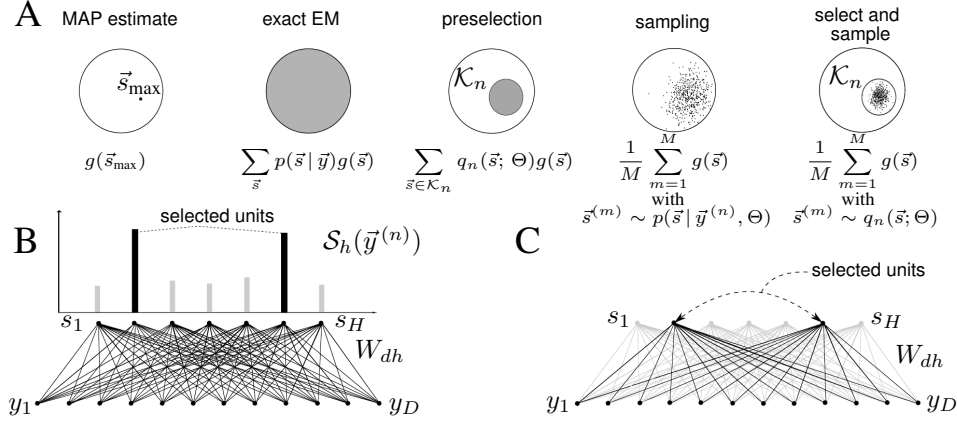

Figure 1: **A** Simplified illustration of the posterior mass and the respective regions each approximation approach uses to compute the expectation values. **B** Graphical model showing each connection $W_{dh}$ between the observed variables $\vec{y}$ and hidden variables $\vec{s}$, and how $H' = 2$ hidden variables/units are selected to form a set $\mathcal{K}_n$. **C** Graphical model resulting from the selection of hidden variables and associated weights $W_{dh}$ (black).

**Select and Sample.** Although preselection is a deterministic approach very different than the stochastic nature of sampling, its formulation as approximation to expectation values (3) allows for a straight-forward combination of both approaches: given a data point, $\vec{y}^{(n)}$, we first approximate the expectation value (3) using the variational distribution $q_n(\vec{s}; \Theta)$ as defined by preselection (2). Second, we approximate the expectations w.r.t. $q_n(\vec{s}; \Theta)$ using sampling. The combined approach is thus given by:

$$\langle g(\vec{s}) \rangle_{p(\vec{s} \mid \vec{y}^{(n)}, \Theta)} \approx \langle g(\vec{s}) \rangle_{q_n(\vec{s}; \Theta)} \approx \frac{1}{M} \sum_{m=1}^{M} g(\vec{s}^{(m)}) \quad \text{with} \quad \vec{s}^{(m)} \sim q_n(\vec{s}; \Theta), \quad (6)$$

where $\vec{s}^{(m)}$ denote samples from the truncated distribution $q_n$. Instead of drawing from a distribution over the entire state space, approximation (6) requires only samples from a potentially very small subspace $\mathcal{K}_n$ (Fig. 1). In the subspace $\mathcal{K}_n$, most of the original probability mass is concentrated in a smaller volume, thus MCMC algorithms perform more efficiently, which results in a smaller space to explore, shorter burn-in times, and a reduced number of required samples. Compared to selection alone, the select and sample approach will represent an increase in efficiency as soon as the number of samples required for a good approximation is less then the number of states in $\mathcal{K}_n$.

## 3  Sparse Coding: An Example Application

We systematically investigate the computational efficiency, performance, and biological plausibility of the select and sample approach in comparison with selection and sampling alone using a sparse coding model of images. The choice of a sparse coding model has numerous advantages. First, it is a non-trivial model that has been extremely well-studied in machine learning research, and for which efficient algorithms exist (e.g., [23, 30]). Second, it has become a standard (albeit somewhat simplistic) model of the organization of receptive fields in primary visual cortex [22, 31, 32]. Here we consider a discrete variant of this model known as Binary Sparse Coding (BSC; [29, 27], also compare [33]), which has binary hidden variables but otherwise the same features as standard sparse coding versions. The generative model for BSC is expressed by

$$p(\vec{s}|\pi) = \prod_{h=1}^{H} \pi^{s_h} (1 - \pi)^{1 - s_h}, \qquad p(\vec{y}|\vec{s}, W, \sigma) = \mathcal{N}(\vec{y}; W\vec{s}, \sigma^2 \mathbb{1}), \quad (7)$$

where $W \in \mathbb{R}^{D \times H}$ denotes the basis vectors and $\pi$ parameterizes the sparsity ($\vec{s}$ and $\vec{y}$ as above). The M-step updates of the BSC learning algorithm (see e.g. [27]) are given by:

$$W^{\text{new}} = \left( \sum_{n=1}^{N} \vec{y}^{(n)} \langle \vec{s} \rangle_{q_n}^{T} \right) \left( \sum_{n=1}^{N} \langle \vec{s}\vec{s}^{T} \rangle_{q_n} \right)^{-1}, \quad (8)$$

$$(\sigma^2)^{\text{new}} = \frac{1}{ND} \sum_n \langle ||\vec{y}^{(n)} - W\vec{s}||^2 \rangle_{q_n}, \quad \pi^{\text{new}} = \frac{1}{N} \sum_n | <\vec{s}>_{q_n} |, \text{ where } |\vec{x}| = \frac{1}{H} \sum_h x_h. \quad (9)$$

The only expectation values needed for the M-step are thus $\langle \vec{s} \rangle_{q_n}$ and $\langle \vec{s}\vec{s}^{T} \rangle_{q_n}$. We will compare learning and inference between the following algorithms:

**BSC$^{\text{exact}}$.** An EM algorithm without approximations is obtained if we use the exact posterior for the expectations: $q_n = p(\vec{s} \mid \vec{y}^{(n)}, \Theta)$. We will refer to this exact algorithm as BSC$^{\text{exact}}$. Although directly computable, the expectation values for BSC$^{\text{exact}}$ require sums over the entire state space, i.e., over $2^H$ terms. For large numbers of latent dimensions, BSC$^{\text{exact}}$ is thus intractable.

**BSC$^{\text{select}}$.** An algorithm that more efficiently scales with the number of hidden dimensions is obtained by applying preselection. For the BSC model we use $q_n$ as given in (3) and $\mathcal{K}_n = \{\vec{s} \mid (\text{for all } h \notin \mathcal{I} : s_h = 0) \text{ or } \sum_h s_h = 1\}$. Note that in addition to states as in (4) we include all states with one non-zero unit (all singletons). Including them avoids EM iterations in the initial phases of learning that leave some basis functions unmodified (see [27]). As selection function $\mathcal{S}_h(\vec{y}^{(n)})$ to define $\mathcal{K}_n$ we use:

$$\mathcal{S}_h(\vec{y}^{(n)}) = (\vec{W}_h^{\mathrm{T}} / ||\vec{W}_h||)\, \vec{y}^{(n)}, \quad \text{with } ||\vec{W}_h|| = \sqrt{\textstyle\sum_{d=1}^{D}(W_{dh})^2}\,. \tag{10}$$

A large value of $\mathcal{S}_h(\vec{y}^{(n)})$ strongly indicates that $\vec{y}^{(n)}$ contains the basis function $\vec{W}_h$ as a component (see Fig. 1C). Note that (10) can be related to a deterministic ICA-like selection of a hidden state $\vec{s}^{(n)}$ in the limit case of no noise (compare [27]). Further restrictions of the state space are possible but require modified M-step equations (see [27, 29]), which will not be considered here.

**BSC$^{\text{sample}}$.** An alternative non-deterministic approach can be derived using Gibbs sampling. Gibbs sampling is an MCMC algorithm which systematically explores the sample space by repeatedly drawing samples from the conditional distributions of the individual hidden dimensions. In other words, the transition probability from the current sample to a new candidate sample is given by $p(s_h^{\text{new}} \mid \vec{s}_{\backslash h}^{\text{current}})$. In our case of a binary sample space, this equates to selecting one random axis $h \in \{1, \ldots, H\}$ and toggling its bit value (thereby changing the binary state in that dimension), leaving the remaining axes unchanged. Specifically, the posterior probability computed for each candidate sample is expressed by:

$$p(s_h = 1 \mid \vec{s}_{\backslash h}, \vec{y}) = \frac{p(s_h = 1, \vec{s}_{\backslash h}, \vec{y})^{\beta}}{p(s_h = 0, \vec{s}_{\backslash h}, \vec{y})^{\beta} + p(s_h = 1, \vec{s}_{\backslash h}, \vec{y})^{\beta}}, \tag{11}$$

where we have introduced a parameter $\beta$ that allows for smoothing of the posterior distribution. To ensure an appropriate mixing behavior of the MCMC chains over a wide range of $\sigma$ (note that $\sigma$ is a model parameter that changes with learning), we define $\beta = \frac{T}{\sigma^2}$, where $T$ is a temperature parameter that is set manually and selected such that good mixing is achieved. The samples drawn in this manner can then be used to approximate the expectation values in (8) to (9) using (5).

**BSC$^{\text{s+s}}$.** The EM learning algorithm given by combining selection and sampling is obtained by applying (6). First note that inserting the BSC generative model into (2) results in:

$$q_n(\vec{s}; \Theta) = \frac{\mathcal{N}(\vec{y}; W\vec{s}, \sigma^2 \mathbb{1}) \operatorname{Bernoulli}_{\mathcal{K}_n}(\vec{s}; \pi)}{\sum_{\vec{s}' \in \mathcal{K}_n} \mathcal{N}(\vec{y}; W\vec{s}', \sigma^2 \mathbb{1}) \operatorname{Bernoulli}_{\mathcal{K}_n}(\vec{s}'; \pi)} \delta(\vec{s} \in \mathcal{K}_n) \tag{12}$$

where $\operatorname{Bernoulli}_{\mathcal{K}_n}(\vec{s}; \pi) = \prod_{h \in \mathcal{I}} \pi^{s_h} (1 - \pi)^{1 - s_h}$. The remainder of the Bernoulli distribution cancels out. If we define $\tilde{\vec{s}}$ to be the binary vector consisting of all entries of $\vec{s}$ of the selected dimensions, and if $\tilde{W} \in \mathbb{R}^{D \times H'}$ contains all basis functions of those selected, we observe that the distribution is equal to the posterior w.r.t. a BSC model with $H'$ instead of $H$ hidden dimensions:

$$p(\tilde{\vec{s}} \mid \vec{y}, \Theta) = \frac{\mathcal{N}(\vec{y}; \tilde{W}\tilde{\vec{s}}, \sigma^2 \mathbb{1}_{H'}) \operatorname{Bernoulli}(\tilde{\vec{s}}; \pi)}{\sum_{\tilde{\vec{s}}'} \mathcal{N}(\vec{y}; \tilde{W}\tilde{\vec{s}}', \sigma^2 \mathbb{1}_{H'}) \operatorname{Bernoulli}(\tilde{\vec{s}}'; \pi)}$$

Instead of drawing samples from $q_n(\vec{s}; \Theta)$ we can thus draw samples from the exact posterior w.r.t. the BSC generative model with $H'$ dimensions. The sampling procedure for BSC$^{\text{sample}}$ can thus be applied simply by ignoring the non-selected dimensions and their associated parameters. For different data points, different latent dimensions will be selected such that averaging over data points can update all model parameters. For selection we again use $\mathcal{S}_h(\vec{y}, \Theta)$ (10), defining $\mathcal{K}_n$ as in (4), where $\mathcal{I}$ now contains the $H'{-}2$ indices $h$ with the highest values of $\mathcal{S}_h(\vec{y}, \Theta)$ and two randomly selected dimensions (drawn from a uniform distribution over all non-selected dimensions). The two randomly selected dimensions fulfill the same purpose as the inclusion of singleton states for BSC$^{\text{select}}$. Preselection and Gibbs sampling on the selected dimensions define an approximation to the required expectation values (3) and result in an EM algorithm referred to as BSC$^{\text{s+s}}$.

**Complexity.** Collecting the number of operations necessary to compute the expectation values for all four BSC cases, we arrive at

$$\mathcal{O}\big(NS(\underbrace{D}_{p(\vec{s},\vec{y})} + \underbrace{1}_{\langle\vec{s}\rangle} + \underbrace{H}_{\langle\vec{s}\vec{s}^T\rangle})\big) \tag{13}$$

where $S$ denotes the number of hidden states that contribute to the calculation of the expectation values. For the approaches with preselection (BSC$^{\text{select}}$, BSC$^{\text{s+s}}$), all the calculations of the expectation values can be performed on the reduced latent space; therefore the $H$ is replaced by $H'$. For BSC$^{\text{exact}}$ this number scales exponentially in $H$: $S^{\text{exact}} = 2^H$, and in in the BSC$^{\text{select}}$ case, it scales exponentially in the number of preselected hidden variables: $S^{\text{select}} = 2^{H'}$. However, for the sampling based approaches (BSC$^{\text{sample}}$ and BSC$^{\text{s+s}}$), the number $S$ directly corresponds to the number of samples to be evaluated and is obtained empirically. As we will show later, $S^{\text{s+s}} = 200 \times H'$ is a reasonable choice for the interval of $H'$ that we investigate in this paper ($1 \leq H' \leq 40$).

## 4 Numerical Experiments

We compare the select and sample approach with selection and sampling applied individually on different data sets: artifical images and natural image patches. For all experiments using the two sampling approaches, we draw 20 independent chains that are initialized at random states in order to increase the mixing of the samples. Also, of the samples drawn per chain, $\frac{1}{3}$ were used to as burn-in samples, and $\frac{2}{3}$ were retained samples.

**Artificial data.** Our first set of experiments investigate the select and sample approach's convergence properties on artificial data sets where ground truth is available. As the following experiments were run on a small scale problem, we can compute the exact data likelihood for each EM step in all four algorithms (BSC$^{\text{exact}}$, BSC$^{\text{select}}$, BSC$^{\text{sample}}$ and BSC$^{\text{s+s}}$) to compare convergence on ground truth likelihood.

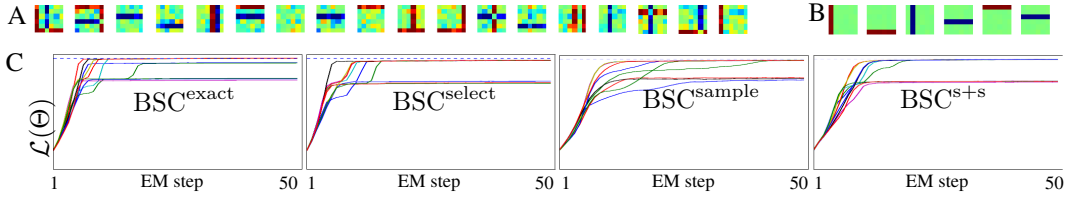

Figure 2: Experiments using artificial bars data with $H = 12$, $D = 6 \times 6$. Dotted line indicates the ground truth log-likelihood value. **A** Random selection of the $N = 2000$ training data points $\vec{y}^{(n)}$. **B** Learned basis functions $W_{dh}$ after a successful training run. **C** Development of the log-likelihood over a period of 50 EM steps for all 4 investigated algorithms.

Data for these experiments consisted of images generated by creating $H = 12$ basis functions $\vec{W}_h^{\text{gt}}$ in the form of horizontal and vertical bars on a $D = 6 \times 6 = 36$ pixel grid. Each bar was randomly assigned to be either positive ($W_{dh}^{\text{gt}} \in \{0.0, 10.0\}$) or negative ($W_{h'd}^{gt} \in \{-10.0, 0.0\}$). $N = 2000$ data points $\vec{y}^{(n)}$ were generated by linearly combining these basis functions (see e.g., [34]). Using a sparseness value of $\pi_{\text{gt}} = \frac{2}{H}$ resulted in, on average, two active bars per data point. According to the model, we added Gaussian noise ($\sigma_{\text{gt}} = 2.0$) to the data (Fig. 2**A**).

We applied all algorithms to the same dataset and monitored the exact likelihood over a period of 50 EM steps (Fig. 2**C**). Although the calculation of the exact likelihood requires $\mathcal{O}(N2^H(D+H))$ operations, this is feasible for such a small scale problem. For models using preselection (BSC$^{\text{select}}$ and BSC$^{\text{s+s}}$), we set $H'$ to 6, effectively halving the number of hidden variables participating in the calculation of the expectation values. For BSC$^{\text{sample}}$ and BSC$^{\text{s+s}}$ we drew 200 samples from the posterior $p(\vec{s}|\vec{y}^{(n)})$ of each data point, as such the number of states evaluated totaled $S^{\text{sample}} = 200 \times H = 2400$ and $S^{\text{s+s}} = 200 \times H' = 1200$, respectively. To ensure an appropriate mixing behavior annealing temperature was set to $T = 50$. In each experiment the basis functions were initialized at the data mean plus Gaussian noise, the prior probability to $\pi_{\text{init}} = \frac{1}{H}$ and the data noise to the variance of the data. All algorithms recover the correct set of bases functions in $> 50\%$ of the trials, and the sparseness prior $\pi$ and the data noise $\sigma$ with high accuracy. Comparing the computational costs of algorithms shows the benefits of preselection already for this small scale problem: while BSC$^{\text{exact}}$ evaluates the expectation values using the full set of $2^H = 4096$ hidden

states, $\text{BSC}^{\text{select}}$ only considers $2^{H'} + (H - H') = 70$ states. The pure sampling based approaches performs 2400 evaluations while $\text{BSC}^{\text{s+s}}$ requires 1200 evaluations.

**Image patches.** We test the select and sample approach on natural image data at a more challenging scale, to include biological plausibility in the demonstration of its applicability to larger scale problems. We extracted $N = 40,000$ patches of size $D = 26 \times 26 = 676$ pixels from the van Hateren image database [31][2], and preprocessed them using a Difference of Gaussians (DoG) filter, which approximates the sensitivity of center-on and center-off neurons found in the early stages of the mammalian visual processing. Filter parameters where chosen as in [35, 28]. For the following experiments we ran 100 EM iterations to ensure proper convergence. The annealing temperature was set to $T = 20$.

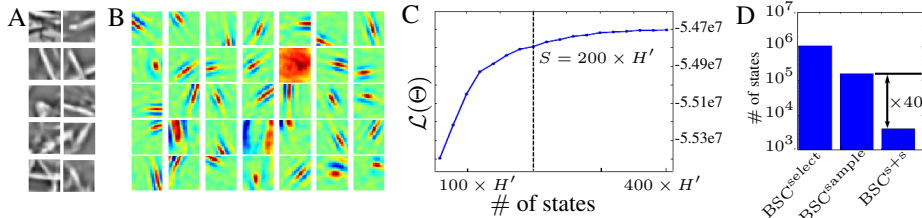

Figure 3: Experiments on image patches with $D = 26 \times 26$, $H = 800$ and $H' = 20$. **A** Random selection of used patches (after DoG preprocessing). **B** Random selection of learned basis functions (number of samples set to 200). **C** End approx. log-likelihood after 100 EM-steps vs. number of samples per data point. **D** Number of states that had to be evaluated for the different approaches.

The first series of experiments investigate the effect of the number of drawn samples on the performance of the algorithm (as measured by the approximate data likelihood) across the entire range of $H'$ values between 12 and 36. We observe with $\text{BSC}^{\text{s+s}}$ that 200 samples per hidden dimension (total states $= 200 \times H'$) are sufficient: the final value of the likelihood after 100 EM steps begins to saturate. Particularly, increasing the number of samples does not increase the likelihood by more than 1%. In Fig. 3**C** we report the curve for $H' = 20$, but the same trend is observed for all other values of $H'$. In another set of experiments, we used this number of samples ($200 \times H$) in the pure sampling case ($\text{BSC}^{\text{sample}}$) in order to monitor the likelihood behavior. We observed two consistent trends: 1) the algorithm was never observed to converge to a high-likelihood solution, and 2) even when initialized at solutions with high likelihood, the likelihood always decreases. This example demonstrates the gains of using select and sample above pure sampling: while $\text{BSC}^{\text{s+s}}$ only needs $200 \times 20 = 4,000$ samples to robustly reach a high-likelihood solutions, by following the same regime with $\text{BSC}^{\text{sample}}$, not only did the algorithm poorly converge on a high-likelihood solution, but it used $200 \times 800 = 160,000$ samples to do so (Fig. 3**D**).

**Large scale experiment on image patches**. Comparison of the above results shows that the most efficient algorithm is obtained by a combination of preselection and sampling, our select and sample approach ($\text{BSC}^{\text{s+s}}$), with no or only minimal effect on the performance of the algorithm – as depicted in Fig. 2 and 3. This efficiency allows for applications to much larger scale problems than would be possible by individual approximation approaches. To demonstrate the efficiency of the combined approach we applied $\text{BSC}^{\text{s+s}}$ to the same image dataset, but with a very high number of observed and hidden dimensions. We extracted from the database $N = 500,000$ patches of size $D = 40 \times 40 = 1,600$ pixels. $\text{BSC}^{\text{s+s}}$ was applied with the number of hidden units set to $H = 1,600$ and with $H' = 34$. Using the same conditions as in the previous experiments (notably $S = 200 \times H' = 64,000$ samples and 100 EM iterations) we again obtain a set of Gabor-like basis functions (see Fig. 4**A**) with relatively very few necessary states (Fig. 4**B**). To our knowledge, the presented results illustrate the largest application of sparse coding with a reasonably complete representation of the posterior.

## 5 Discussion

We have introduced a novel and efficient method for unsupervised learning in probabilistic models – one which maintains a complex representation of the posterior for problems consistent with

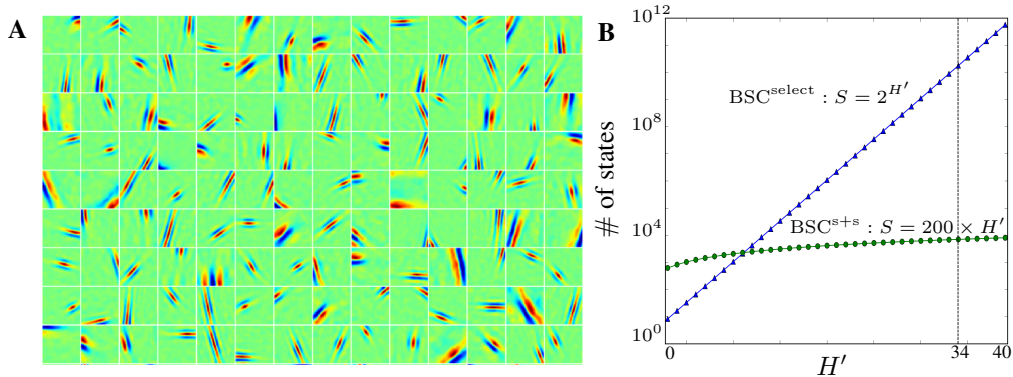

Figure 4: **A** Large-scale application of $\mathrm{BSC}^{\mathrm{s+s}}$ with $H' = 34$ to image patches ($D = 40 \times 40 = 1600$ pixels and $H = 1600$ hidden dimensions). A random selection of the inferred basis functions is shown (see Suppl for all basis functions and model parameters). **B** Comparison the of computational complexity: $\mathrm{BSC}^{\mathrm{select}}$ scales exponentially with $H'$ whereas $\mathrm{BSC}^{\mathrm{s+s}}$ scales linearly. Note the large difference at $H' = 34$ as used in **A**.

real-world scales. Furthermore, our approach is biologically plausible and models how the brain can make sense of its environment for large-scale sensory inputs. Specifically, the method could be implemented in neural networks using two mechanisms, both of which have been independently suggested in the context of a statistical framework for perception: feed-forward preselection [27], and sampling [12, 13, 3]. We showed that the two seemingly contrasting approaches can be combined based on their interpretation as approximate inference methods, resulting in a considerable increase in computational efficiency (e.g., Figs. 3-4).

We used a sparse coding model of natural images – a standard model for neural response properties in V1 [22, 31] – in order to investigate, both numerically and analytically, the applicability and efficiency of the method. Comparisons of our approach with exact inference, selection alone, and sampling alone showed a very favorable scaling with the number of observed and hidden dimensions. To the best of our knowledge, the only other sparse coding implementation that reached a comparable problem size ($D = 20 \times 20, H = 2\,000$) assumed a Laplace prior and used a MAP estimation of the posterior [23]. However, with MAP estimations, basis functions have to be rescaled (compare [22]) and data noise or prior parameters cannot be inferred (instead a regularizer is hand-set). Our method does not require any of these artificial mechanisms because of its rich posterior representation. Such representations are, furthermore, crucial for inferring all parameters such as data noise and sparsity (learned in all of our experiments), and to correctly act when faced with uncertain input [2, 8, 3]. Concretely, we used a sparse coding model with binary latent variables. This allowed for a systematic comparison with exact EM for low-dimensional problems, but extension to the continuous case should be straight-forward. In the model, the selection step results in a simple, local and neurally plausible integration of input data, given by (10). We used this in combination with Gibbs sampling, which is also neurally plausible because neurons can individually sample their next state based on the current state of the other neurons, as transmitted through recurrent connections [15]. The idea of combining sampling with feed-forward mechanisms has previously been explored, but in other contexts and with different goals. Work by Beal [36] used variational approximations as proposal distributions within importance sampling, and Zhu et al. [37] guided a Metropolis-Hastings algorithm by a data-driven proposal distribution. Both approaches are different from selecting subspaces prior to sampling and are more difficult to link to neural feed-forward sweeps [18, 21].

We expect the select and sample strategy to be widely applicable to machine learning models whenever the posterior probability masses can be expected to be concentrated in a small sub-space of the whole latent space. Using more sophisticated preselection mechanisms and sampling schemes could lead to a further reduction in computational efforts, although the details will depend in general on the particular model and input data.

**Acknowledgements.** We acknowledge funding by the German Research Foundation (DFG) in the project LU 1196/4-1 (JL), by the German Federal Ministry of Education and Research (BMBF), project 01GQ0840 (JAS, JB, ASS), by the Swartz Foundation and the Swiss National Science Foundation (PB). Furthermore, support by the Physics Dept. and the Center for Scientific Computing (CSC) in Frankfurt are acknowledged.

## Footnotes

[1]In the case of continuous variables the sum is replaced by an integral. For a hierarchical model, the prior distribution $p(\vec{s} \,|\, \Theta)$ may be subdivided hierarchically into different sets of variables.

[2]We restricted the set of images to 900 images without man-made structures (see Fig 3**A**). The brightest 2% of the pixels were clamped to the max value of the remaining 98% (reducing influences of light-reflections)

# References

[1] P. Dayan and L. F. Abbott. *Theoretical Neuroscience*. MIT Press, Cambridge, 2001.

[2] R. P. N. Rao, B. A. Olshausen, and M. S. Lewicki. *Probabilistic Models of the Brain: Perception and Neural Function*. MIT Press, 2002.

[3] J. Fiser, P. Berkes, G. Orban, and M. Lengye. Statistically optimal perception and learning: from behavior to neural representations. *Trends in Cognitive Sciences*, 14:119–130, 2010.

[4] M. D. Ernst and M. S. Banks. Humans integrate visual and haptic information in a statistically optimal fashion. *Nature*, 415:419–433, 2002.

[5] Y. Weiss, E. P. Simoncelli, and E. H. Adelson. Motion illusions as optimal percepts. *Nature Neuroscience*, 5:598–604, 2002.

[6] K. P. Kording and D. M. Wolpert. Bayesian integration in sensorimotor learning. *Nature*, 427:244–247, 2004.

[7] J. M. Beck, W. J. Ma, R. Kiani, T. Hanksand A. K. Churchland, J. Roitman, M. N.. Shadlen, P. E. Latham, and A. Pouget. Probabilistic population codes for bayesian decision making. *Neuron*, 60(6), 2008.

[8] J. Trommershäuser, L. T. Maloney, and M. S. Landy. Decision making, movement planning and statistical decision theory. *Trends in Cognitive Science*, 12:291–297, 2008.

[9] P. Berkes, G. Orban, M. Lengyel, and J. Fiser. Spontaneous cortical activity reveals hallmarks of an optimal internal model of the environment. *Science*, 331(6013):83–87, 2011.

[10] W. J. Ma, J. M. Beck, P. E. Latham, and A. Pouget. Bayesian inference with probabilistic population codes. *Nature Neuroscience*, 9:1432–1438, 2006.

[11] R. Turner, P. Berkes, and J. Fiser. Learning complex tasks with probabilistic population codes. In *Frontiers in Neuroscience*, 2011. Comp. and Systems Neuroscience 2011.

[12] T. S. Lee and D. Mumford. Hierarchical Bayesian inference in the visual cortex. *Journal of the Optical Society of America A*, 20(7):1434–1448, 2003.

[13] P. O. Hoyer and A. Hyvarinen. Interpreting neural response variability as Monte Carlo sampling from the posterior. In *Adv. Neur. Inf. Proc. Syst. 16*, pages 293–300. MIT Press, 2003.

[14] E. Vul, N. D. Goodman, T. L. Griffiths, and J. B. Tenenbaum. One and done? Optimal decisions from very few samples. In *31st Annual Meeting of the Cognitive Science Society*, 2009.

[15] P. Berkes, R. Turner, and J. Fiser. The army of one (sample): the characteristics of sampling-based probabilistic neural representations. In *Frontiers in Neuroscience*, 2011. Comp. and Systems Neuroscience 2011.

[16] F. Rosenblatt. The perceptron: A probabilistic model for information storage and organization in the brain. *Psychological Review*, 65(6), 1958.

[17] M. Riesenhuber and T. Poggio. Hierarchical models of object recognition in cortex. *Nature Neuroscience*, 211(11):1019 – 1025, 1999.

[18] V. A. F.. Lamme and P. R. Roelfsema. The distinct modes of vision offered by feedforward and recurrent processing. *Trends in Neurosciences*, 23(11):571 – 579, 2000.

[19] A. Yuille and D. Kersten. Vision as bayesian inference: analysis by synthesis? *Trends in Cognitive Sciences*, 10(7):301–308, 2006.

[20] G. E. Hinton, P. Dayan, B. J. Frey, and R. M. Neal. The 'wake-sleep' algorithm for unsupervised neural networks. *Science*, 268:1158 – 1161, 1995.

[21] E. Körner, M. O. Gewaltig, U. Körner, A. Richter, and T. Rodemann. A model of computation in neocortical architecture. *Neural Networks*, 12:989 – 1005, 1999.

[22] B. A. Olshausen and D. J. Field. Emergence of simple-cell receptive field properties by learning a sparse code for natural images. *Nature*, 381:607–609, 1996.

[23] H. Lee, A. Battle, R. Raina, and A. Ng. Efficient sparse coding algorithms. *NIPS*, 20:801–808, 2007.

[24] Y. LeCun. Backpropagation applied to handwritten zip code recognition.

[25] M. Riesenhuber and T. Poggio. How visual cortex recognizes objects: The tale of the standard model. 2002.

[26] T. S. Lee and D. Mumford. Hierarchical bayesian inference in the visual cortex. *J Opt Soc Am A Opt Image Sci Vis*, 20(7):1434–1448, July 2003.

[27] J. Lücke and J. Eggert. Expectation Truncation And the Benefits of Preselection in Training Generative Models. *Journal of Machine Learning Research*, 2010.

[28] G. Puertas, J. Bornschein, and J. Lücke. The maximal causes of natural scenes are edge filters. *NIPS*, 23, 2010.

[29] M. Henniges, G. Puertas, J. Bornschein, J. Eggert, and J. Lücke. Binary sparse coding. *Latent Variable Analysis and Signal Separation*, 2010.

[30] J. Mairal, F. Bach, J. Ponce, and G. Sapiro. Online learning for matrix factorization and sparse coding. *The Journal of Machine Learning Research*, 11, 2010.

[31] J. Hateren and A. Schaaf. Independent Component Filters of Natural Images Compared with Simple Cells in Primary Visual Cortex. *Proc Biol Sci*, 265(1394):359–366, 1998.

[32] D. L. Ringach. Spatial Structure and Symmetry of Simple-Cell Receptive Fields in Macaque Primary Visual Cortex. *J Neurophysiol*, 88:455–463, 2002.

[33] M. Haft, R. Hofman, and V. Tresp. Generative binary codes. *Pattern Anal Appl*, 6(4):269–284, 2004.

[34] P. O. Hoyer. Non-negative sparse coding. *Neural Networks for Signal Processing XII: Proceedings of the IEEE Workshop*, pages 557–565, 2002.

[35] J. Lücke. Receptive Field Self-Organization in a Model of the Fine Structure in V1 Cortical Columns. *Neural Computation*, 2009.

[36] M. J. Beal. *Variational Algorithms for Approximate Bayesian Inference*. PhD thesis, Gatsby Computational Neuroscience Unit, University College London., 2003.

[37] Z. Tu and S. C. Zhu. Image Segmentation by Data-Driven Markov Chain Monte Carlo. *PAMI*, 24(5):657–673, 2002.

